# Nonrigid Structure from Motion in Trajectory Space

**Ijaz Akhter**
LUMS School of Science and Engineering
Lahore, Pakistan
akhter@lums.edu.pk

**Yaser Sheikh**
Carnegie Mellon University
Pittsburgh, PA, USA
yaser@cs.cmu.edu

**Sohaib Khan**
LUMS School of Science and Engineering
Lahore, Pakistan
sohaib@lums.edu.pk

**Takeo Kanade**
Carnegie Mellon University
Pittsburgh, PA, USA
tk@cs.cmu.edu

## Abstract

Existing approaches to nonrigid structure from motion assume that the instantaneous 3D shape of a deforming object is a linear combination of basis shapes, which have to be estimated anew for each video sequence. In contrast, we propose that the evolving 3D structure be described by a linear combination of basis trajectories. The principal advantage of this approach is that we do not need to estimate any basis vectors during computation. We show that *generic* bases over trajectories, such as the Discrete Cosine Transform (DCT) basis, can be used to compactly describe most real motions. This results in a significant reduction in unknowns, and corresponding stability in estimation. We report empirical performance, quantitatively using motion capture data, and qualitatively on several video sequences exhibiting nonrigid motions including piece-wise rigid motion, partially nonrigid motion (such as a facial expression), and highly nonrigid motion (such as a person dancing).

## 1 Introduction

Nonrigid structure from motion is the process of recovering the time varying 3D coordinates of points on a deforming object from their 2D locations in an image sequence. Factorization approaches, first proposed for recovering rigid structure by Tomasi and Kanade in [1], were extended to handle nonrigidity in the seminal paper by Bregler *et al.* in [2]. The key idea in [2] is that observed shapes can be represented as a linear combination of a compact set of basis shapes. Each instantaneous structure, such as the mouth of a smiling actor shown in Figure 1(a), is expressed as a point in the linear space of shapes spanned by the shape basis. A number of approaches that develop the use of shape basis have subsequently been proposed, including [3, 4, 5]. Since the space of spatial deformations is highly object specific, the shape basis need to be estimated anew for each video sequence. The shape basis of a mouth smiling, for instance, cannot be recycled to compactly represent a person walking.

In this paper, we posit that representing nonrigid structure as a combination of basis shapes is one of two ways of looking at the space-time structure induced by $P$ points seen across $F$ frames. Instead of a shape space representation, we propose looking across time, representing the time-varying structure of a nonrigid object as a linear combination of a set of basis *trajectories*, as illustrated in Figure 1(b). The principal advantage of taking this "lateral" approach arises from the fact that compact representation in trajectory space is better motivated physically than compact representation in shape space. To see this, consider a deformable object being acted upon by a force. The extent of its deformation is limited by the force that can be applied. Hence, a tree swaying in the wind or a person walking cannot arbitrarily and randomly deform; the trajectories of their points are a function of the speed of the wind and the flexing of muscles respectively. Deformations are, there-

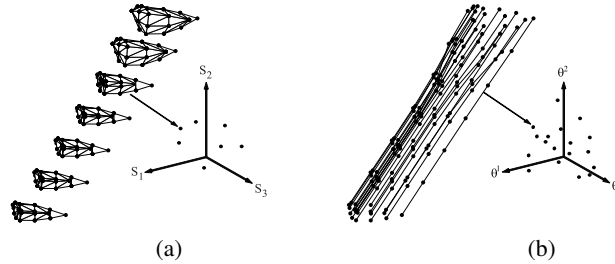

(a)                                   (b)

Figure 1: 3D points on a smiling mouth: a comparison of shape and trajectory space. (a) In approaches that represent the time varying structure in shape space, all 3D points observed at one time instant are projected onto a single point in the shape space. $S_1, S_2, \cdots, S_k$ each represent a shape basis vector. (b) In our approach, we represent the time varying structure in trajectory space, where a 3D point's trajectory over time is projected to a single point in the trajectory space. $\theta^1, \theta^2, \cdots, \theta^k$ each represent a trajectory basis vector. $P$ points observed across $F$ frames are expressed as $F$ projected points in shape space and $P$ points in trajectory space.

fore, constrained by the physical limits of actuation to remain incremental, not random, across time. Since this property is, to a large degree, ubiquitous, basis can be defined in trajectory that are *object independent*.

We show that while the inherent representative power of both shape and trajectory projections of structure data are equal (a duality exists), the significant reduction in number of unknowns that results from knowing the basis *apriori* allows us to handle much more nonrigidity of deformation than state of the art methods, like [4] and [5]. In fact, most previous results consider deformations which have a large rigid component, such as talking-head videos or the motion of a swimming shark. To the best of our knowledge, we are the first to show reasonable reconstructions of highly nonrigid motions from a single video sequence without making object specific assumptions. For all results, we use the same trajectory basis, the Discrete Cosine Transform (DCT) basis, underlining the generic nature of the trajectory space representation. A useful byproduct of this approach is that structure is automatically compressed for compact transmission without the need for *post facto* compression or the overhead transmission of object specific basis.

## 2   Related work

If deformation of a 3D scene is unconstrained, the structure observed in each image would be independent of those in other images. In this case, recovering structure from motion is ill-posed, equivalent to finding 3D structure from a single 2D image at each time instant. To make nonrigid structure recovery tractable, some consistency in the deformation of structure has to be imposed. One early measure of consistency that was applied assumes that the scene consists of multiple rigid objects which are moving independently [6, 7, 8]. However, the first *general* solution to the problem of nonrigid structure recovery was introduced by Bregler *et al.* in [2], approximating the structure at each time instant as a linear combination of basis shapes. They recovered the structure, the shape basis and the camera rotations simultaneously, by exploiting orthonormality constraints of the rotation matrices. Xiao *et al.* [4] showed that these orthonormality constraints alone lead to ambiguity in the solution, and introduced additional constraints to remove ambiguity. In [9] Xiao *et al.* proposed a rank deficient basis. Other extensions of the work by Bregler *et al.* include [10] which improved the numerical stability of the estimation process and [3] which introduced a Gaussian prior on the shape coefficients. Common to all of these approaches is that results are shown on objects which have a significant number of points that move rigidly, such as faces. Some approaches, such as [11] make explicit use of this fact to initialize rotation matrices, while others favor such sequences for stability in estimation.

In contrast to this entire corpus of work, which approximate structure by a shape basis, we propose a new representation of time varying structure, as a collection of trajectories. We not only demonstrate that a compact trajectory space can be defined, but also that the basis of this trajectory space can be pre-defined, removing a large number of unknowns from the estimation process altogether. The duality of spatial and temporal representations has been hinted at earlier in literature. Shashua [12] discusses the duality of the *joint image space* and the *joint point space* in the context of multiview geometry. Zelnik-Manor and Irani [13] have exploited a similar duality for an alternate approach to

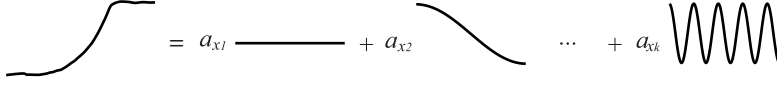

Figure 2: As described in Equation 3, each trajectory is represented as a linear combination of $k$ predefined basis trajectories. In this paper, we use DCT basis to compactly represent trajectories.

segmenting video sequences. Ours is the first paper to use this dual representation in the structure from motion problem, and to note that a generic basis can be defined in trajectory space which compactly represents most real trajectories.

## 3 Representing Nonrigid Structure

The structure at a time instant $t$ can be represented by arranging the 3D locations of the $P$ points in a matrix $S(t) \in \mathbb{R}^{3 \times P}$,

$$S(t) = \left[ \begin{array}{ccc} X_{t1} & & X_{tP} \\ Y_{t1} & \cdots & Y_{tP} \\ Z_{t1} & & Z_{tP} \end{array} \right].$$

The complete time varying structure can be represented by concatenating these instantaneous structures as $\mathbf{S}_{3F \times P} = [S(1)^T \ S(2)^T \ \cdots \ S(F)^T]^T$. In [2], each instantaneous shape matrix $S(t)$ is approximated as a linear combination of basis shapes,

$$S(t) = \sum_j c_j(t) S^j, \tag{1}$$

where $S^j \in \mathbb{R}^{3 \times P}$ is a basis shape and $c_j(t)$ is the coefficient of that basis shape. If the set of observed structures can be compactly expressed in terms of $k$ such basis shapes, $\mathbf{S}$ has a rank of at most $3k$. This rank constraint can be restated by rearrangement of $\mathbf{S}$ as the following rank $k$ matrix,

$$\mathbf{S}^* = \left[ \begin{array}{ccccccccc} X_{11} & \cdots & X_{1P} & Y_{11} & \cdots & Y_{1P} & Z_{11} & \cdots & Z_{1P} \\ \vdots & & \vdots & \vdots & & \vdots & \vdots & & \vdots \\ X_{F1} & \cdots & X_{FP} & Y_{F1} & \cdots & Y_{FP} & Z_{F1} & \cdots & Z_{FP} \end{array} \right]. \tag{2}$$

The row space of this matrix corresponds to the shape space. Since the row and column space of a matrix are of equal dimension, it follows that the columns of $\mathbf{S}^*$ are also spanned by $k$ vectors. We call the column space of this matrix the *trajectory space* and note that it enjoys a dual relationship with the shape space. Specifically, if the time varying shape of an object can be expressed by a minimum of $k$ shape basis, then there exist exactly $k$ trajectory basis vectors that can represent the same time varying shape.

To represent the time varying structure in terms of trajectory basis, we consider the structure as a set of trajectories, $T(i) = [T_x(i)^T \ T_y(i)^T \ T_z(i)^T]^T$, (see Figure 1(b)) where $T_x(i) = [X_{1i}, \cdots, X_{Fi}]^T$, $T_y(i) = [Y_{1i}, \cdots, Y_{Fi}]^T$, $T_z(i) = [Z_{1i}, \cdots, Z_{Fi}]^T$ are the $x$, $y$, and $z$ coordinates of the $i$th trajectory. As illustrated in Figure 2, we describe each trajectory as a linear combination of basis trajectory,

$$T_x(i) = \sum_{j=1}^{k} a_{xj}(i) \theta^j, \quad T_y(i) = \sum_{j=1}^{k} a_{yj}(i) \theta^j, \quad T_z(i) = \sum_{j=1}^{k} a_{zj}(i) \theta^j, \tag{3}$$

where $\theta^j \in \mathbb{R}^F$ is a trajectory basis vector and $a_{xj}(i), a_{yj}(i)$ and $a_{zj}(i)$ are the coefficients corresponding to that basis vector. The time varying structure matrix can then be factorized into an inverse projection matrix and coefficient matrix as $\mathbf{S}_{3F \times P} = \mathbf{\Theta}_{3F \times 3k} \mathcal{A}_{3k \times P}$, where $\mathcal{A} = [\mathbf{A}_x^T \mathbf{A}_y^T \mathbf{A}_z^T]^T$ and

$$\mathbf{A}_x = \left( \begin{array}{ccc} a_{x1}(1) & \cdots & a_{x1}(P) \\ \vdots & & \vdots \\ a_{xk}(1) & \cdots & a_{xk}(P) \end{array} \right), \mathbf{\Theta} = \left( \begin{array}{ccccc} \theta_1^T & & & & \\ & \theta_1^T & & & \\ & & \theta_1^T & & \\ & & & \ddots & \\ \theta_F^T & & & & \\ & \theta_F^T & & & \\ & & \theta_F^T & & \end{array} \right), \tag{4}$$

Here $\theta_i$ represents a truncated basis for transformation from coefficient space to original space. The principal benefit of the trajectory space representation is that a basis can be pre-defined that can compactly approximate most real trajectories. A number of bases such as the Hadamard Transform basis, the Discrete Fourier Transform basis, and the Discrete Wavelet Transform basis can all compactly represent trajectories in an object independent way. In this paper, we use the Discrete Cosine Transform basis set to generate $\mathbf{\Theta}$ (shown in Figure 2) for all reconstructions results shown. The efficacy of the DCT basis has been demonstrated for compressing motion capture data, [14], and has been effective in our experiments as well.

## 4   Nonrigid Structure and Motion Factorization

The measured 2D trajectories are contained in a $2F \times P$ measurement matrix $\mathbf{W}$, containing the location of $P$ image points across $F$ frames,

$$\mathbf{W} = \begin{pmatrix} u_{11} & \ldots & u_{1P} \\ v_{11} & \ldots & v_{1P} \\ \vdots & & \vdots \\ u_{F1} & \ldots & u_{FP} \\ v_{F1} & \ldots & v_{FP} \end{pmatrix}.$$

This measurement matrix can be decomposed as $\mathbf{W} = \mathcal{R}\mathbf{S}$ where $\mathcal{R}$ is a $2F \times 3F$ matrix,

$$\mathcal{R} = \begin{pmatrix} \mathbf{R}_1 & & \\ & \ddots & \\ & & \mathbf{R}_F \end{pmatrix},$$

and $\mathbf{R}_t$ is a $2 \times 3$ orthographic projection matrix. In the previous section we showed that $\mathbf{S} = \mathbf{\Theta}\mathcal{A}$, as a result we can further factorize $\mathbf{W}$ as

$$\mathbf{W} = \mathcal{R}\mathbf{\Theta}\mathcal{A} = \mathbf{\Lambda}\mathcal{A}, \tag{5}$$

where $\mathbf{\Lambda} = \mathcal{R}\mathbf{\Theta}$. Since $\mathbf{\Lambda}$ is a $3F \times 3k$ matrix, the rank of matrix $\mathbf{W}$ will be at most $3k$. This is a dual property to the rank constraint defined by [2]. We can use SVD to factorize $\mathbf{W}$ as,

$$\mathbf{W} = \hat{\mathbf{\Lambda}}\hat{\mathcal{A}}.$$

In general, the matrix $\hat{\mathbf{\Lambda}}$ and $\hat{\mathcal{A}}$ will not be equal to $\mathbf{\Lambda}$ and $\mathcal{A}$ respectively, because the above factorization is not unique. For any invertible $3k \times 3k$ matrix $\mathbf{Q}$, $\hat{\mathbf{\Lambda}}\mathbf{Q}$ and $\mathbf{Q}^{-1}\hat{\mathcal{A}}$ are also valid factorizations. Therefore, to recover metric structure we need to estimate the rectification matrix $\mathbf{Q}$ such that the following equations hold true,

$$\mathbf{\Lambda} = \hat{\mathbf{\Lambda}}\mathbf{Q}, \quad \mathcal{A} = \mathbf{Q}^{-1}\hat{\mathcal{A}}. \tag{6}$$

## 5   Metric Upgrade

The problem of recovering the rotation and structure is reduced to estimating the rectification matrix $\mathbf{Q}$. The elements of matrix $\mathbf{\Lambda}$ are,

$$\mathbf{\Lambda} = \begin{pmatrix} r_1^1\theta_1^T & r_2^1\theta_1^T & r_3^1\theta_1^T \\ r_4^1\theta_1^T & r_5^1\theta_1^T & r_6^1\theta_1^T \\ & \vdots & \\ r_1^F\theta_F^T & r_2^F\theta_F^T & r_3^F\theta_F^T \\ r_4^F\theta_F^T & r_5^F\theta_F^T & r_6^F\theta_F^T \end{pmatrix}.$$

Instead of estimating the whole matrix $\mathbf{Q}$, to rectify $\hat{\Lambda}$ and $\hat{\mathcal{A}}$ it is sufficient to estimate only three columns of $\mathbf{Q}$. Let us define $\mathbf{Q}_{|||}$ to be the first, $k+1^{st}$ and $2k+1^{st}$ columns of the matrix $\mathbf{Q}$. From Equation 6, if we just use $\mathbf{Q}_{|||}$ instead of $\mathbf{Q}$, we get

$$\hat{\mathbf{\Lambda}}\mathbf{Q}_{|||} = \begin{pmatrix} \theta_{1,1}R_1 \\ \vdots \\ \theta_{F,1}R_F \end{pmatrix}. \tag{7}$$

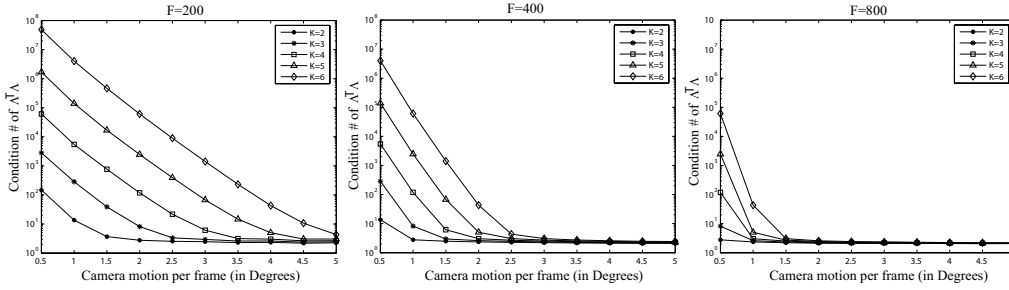

Figure 3: Effect of increasing camera motion on reconstruction stability. Reconstruction stability is measured in terms of condition number of matrix $\mathbf{\Lambda}^T\mathbf{\Lambda}$ with different values of $k$ and different values of $F$. Synthetic rotations were generated by revolving the camera around the $z$-axis and camera motion was measured in terms of the angle the camera moved per frame.

This equation shows that the unknowns in matrix $\mathbf{Q}_{|||}$ can be found by exploiting the fact that $R_i$ is a truncated rotation matrix (as was done in [1]). Specifically, if $\hat{\mathbf{\Lambda}}_{2i-1:2i}$ denotes the two rows of matrix $\hat{\mathbf{\Lambda}}$ at positions $2i-1$ and $2i$, then we have

$$\hat{\mathbf{\Lambda}}_{2i-1:2i}\mathbf{Q}_{|||}\mathbf{Q}_{|||}^T\hat{\mathbf{\Lambda}}_{2i-1:2i}^T = \theta_{i,1}^2 I_{2\times 2}, \tag{8}$$

where $I_{2\times 2}$ is an identity matrix, giving three indepedent constraints for each image $i$. Therefore for $F$ frames, we have $3F$ constraints and $9k$ unknowns in $\mathbf{Q}_{|||}$. Hence at least $3k$ non-degenerate images are required to estimate $\mathbf{Q}_{|||}$. Once $\mathbf{Q}_{|||}$ has been computed, using a nonlinear minimization routine (e.g. Levenberg Marquardt), we can estimate the rotation matrices, and therefore $\mathcal{R}$, using Equation 7.

Once $\mathcal{R}$ is known, it can be multiplied with the (known) DCT basis matrix $\mathbf{\Theta}_{3F\times 3k}$ to recover the matrix $\mathbf{\Lambda}_{2F\times 3k} = \mathcal{R}_{2F\times 3F}\mathbf{\Theta}_{3F\times 3k}$. The coefficients can then be estimated by solving the following overconstrained linear system of equations,

$$\mathbf{\Lambda}_{2F\times 3k}\hat{\mathcal{A}}_{3k\times P} = \mathbf{W}_{2F\times P}. \tag{9}$$

## 6   Results

The proposed algorithm has been validated quantitatively on motion capture data over different actions and qualitatively on video data. We have tested the approach extensively on highly nonrigid human motion like volleyball digs, handstands, karate moves and dancing. Figure 4 shows a few sample reconstructions of different actors. As mentioned earlier, we choose DCT as the basis for the trajectory space. In subsequent experiments, we compare our approach with [5] and [9] (we use code kindly provided by the respective authors). The results, data and the code used to produce the results are all shared at http://cvlab.lums.edu.pk/nrsfm.

In nonrigid structure from motion, the key relationship that determines successful reconstruction is the one between the degree of deformation of the object, measured by the number of basis $k$ required to approximate it and the degree of camera motion. To test the relationship between $k$, camera motion and reconstruction stability, we constructed $\mathbf{\Lambda}$ matrices using different values of $k$ and synthetic rotations around the $z$-axis, at various magnitudes of motion per frame. In Figure 3, the reconstruction stability, measured by the condition number of $\mathbf{\Lambda}^T\mathbf{\Lambda}$, is shown as $k$ is varied between 2 and 6, for 200, 400, and 800 frames (at different angular velocities per frame). The plots confirm intuition: the smaller the degree of object deformation and the larger the camera motion, the more stable reconstruction tends to be.

For quantitative evaluation of reconstruction accuracy we used the drink, pickup, yoga, stretch, and dance actions from the CMU Mocap database, and the shark dataset of [3]. Multiple rigid body data was generated by simulation of points on rigidly moving cubes. We generated synthetic camera rotations and projected 3D data using these rotations to get image observations. The camera rotation for the Mocap datasets was 5 degrees per frame and 2 degrees per frame for the multi-body

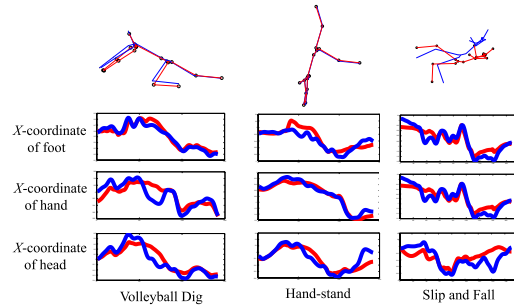

Figure 4: Simultaneous reconstruction accuracy for three actors. The $X$-coordinate trajectories for three different points on the actors is shown. The approximation error introduced by DCT projection has a smoothing impact on the reconstruction. Red lines indicate ground truth data and blue lines indicate reconstructed data.

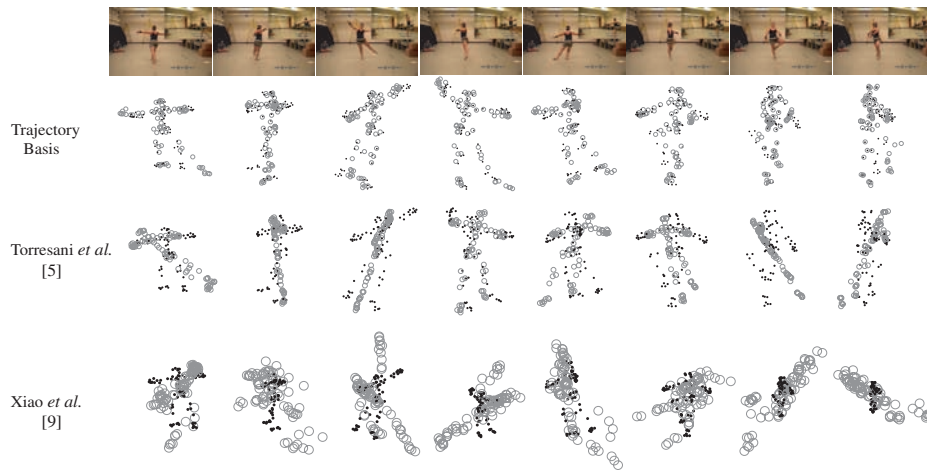

Figure 5: The dance sequence from the CMU mocap database. The black dots are the ground truth points while the gray circles are the reconstructions by the three methods respectively.

sequence. We did not rotate the camera for the dance and shark sequences, since the object itself was rotating in these sequences. In obtaining the results discussed below, $k$ was chosen to provide the best reconstructions, the value varying between 2 and 13 depending on the length of the sequence and the nonrigidity of motion. We normalize the structure, so that the average standard deviation of the structure matrix $\mathbf{S}$ becomes equal to unity (to make comparison of error across datasets more meaningful).

Table 1 shows a quantitative comparison of our method with the shape basis approach of Torresani *et al.* [5] and Xiao and Kanade [9]. This table shows both the camera rotation estimation error and structure reconstruction error. The estimated structure is valid up to a 3D rotation and translation and the estimated rotations also have a 3D rotation ambiguity. We therefore align them for error measurement. Procrustes analysis was used for aligning camera rotations and the 3D structure. The error measure for camera rotations was the average Frobenius norm difference between the original camera rotation and the estimated camera rotation. For structure evaluation we compute the per frame mean squared error between original 3D points and the estimated 3D points.

Finally, to test the proposed approach on real data, we used a face sequence from the PIE dataset, a sequence from the movie "The Matrix", a sequence capturing two rigidly moving cubes and a sequence of a toy dinosaur moving nonrigidly. For the last three sequences, the image points were tracked in a semi-automatic manner, using the approach proposed in [15] with manual correction. We show the resulting reconstructions in Figure 6, and compare against the reconstructions obtained from Torresani *et al.* [5] and Xiao and Kanade [9].

Table 1: The quantitative comparison of proposed algorithm with the techniques described in Xiao and Kanade [9] and Torresani *et al.* [5]. The $E_{rot}$ is the average Frobenius difference between original rotations and aligned estimated rotations, and $E_\Delta$ is the average distance between original 3D points and aligned reconstructed points

| | Trajectory Bases | | Torresani's EM-Gaussian | | Xiao's Shape Bases | |
|---|---|---|---|---|---|---|
| **Datset** | $E_{rot}$ | $E_\Delta$ | $E_{rot}$ | $E_\Delta$ | $E_{rot}$ | $E_\Delta$ |
| DRINK | 5.8E-03 | 2.50E-02 | 0.2906 | 0.3393 | 0.3359 | 3.5186 |
| PICKUP | 1.55E-01 | 2.37E-01 | 0.4277 | 0.5822 | 0.4687 | 3.3721 |
| YOGA | 1.06E-01 | 1.62E-01 | 0.8089 | 0.8097 | 1.2014 | 7.4935 |
| STRETCH | 5.49E-02 | 1.09E-01 | 0.7594 | 1.1111 | 0.9489 | 4.2415 |
| MULTIRIGID | 1.96E-08 | 4.88E-02 | 0.1718 | 2.5902 | 0.0806 | 11.7013 |
| DANCE | NA | 2.96E-01 | NA | 0.9839 | NA | 2.9962 |
| SHARK | NA | 3.12E-01 | NA | 0.1086 | NA | 0.4772 |

# 7    Conclusion

We describe an algorithm to reconstruct nonrigid structure of an object from 2D trajectories of points across a video sequence. Unlike earlier approaches that require an object-specific shape basis to be estimated for each new video sequence, we demonstrate that a generic trajectory basis can be defined that can compactly represent the motion of a wide variety of real deformations. Results are shown using the DCT basis to recover structures of piece-wise rigid motion, facial expressions, actors dancing, walking, and doing yoga. Our experiments show that there is a relationship between camera motion, degree of object deformation, and reconstruction stability. We observe that as the motion of the camera increases with respect to the degree of deformation, the reconstruction stability increases. Future directions of research include experimenting with different unitary transform bases to verify that DCT basis are, in fact, the best generic basis to use, and developing a synergistic approach to use both shape and trajectory bases concurrently.

# 8    Acknowledgements

This research was partially supported by a grant from the Higher Education Commission of Pakistan. The authors would like to acknowledge Fernando De La Torre for useful discussions. We further thank J. Xiao, L. Agapito, I. Matthews and L. Torresani for making their code or data available to us. The motion capture data used in this project was obtained from `http://mocap.cs.cmu.edu`.

# References

[1] C. Tomasi and T. Kanade. Shape and motion from image streams under orthography: A factorization method. *IJCV*, 9:137–154, 1992.

[2] C. Bregler, A. Hertzmann, and H. Biermann. Recovering non-rigid 3D shape from image streams. *CVPR*, 2:690–696, 2000.

[3] L. Torresani, A. Hertzmann, and C. Bregler. Learning non-rigid 3D shape from 2D motion. *NIPS*, 2005.

[4] J. Xiao, J. Chai, and T. Kanade. A closed form solution to non-rigid shape and motion recovery. *IJCV*, 67:233–246, 2006.

[5] L. Torresani, A. Hertzmann, and C. Bregler. Nonrigid structure-from motion: Estimating shape and motion with hierarchical priors. *PAMI*, 30(5):878–892, May 2008.

[6] J.P. Costeira and T. Kanade. A multibody factorization method for independently moving objects. *IJCV*, 49:159–179, 1998.

[7] M. Han and T. Kanade. Reconstruction of a scene with multiple linearly moving objects. *IJCV*, 59:285–300, 2004.

[8] A. Gruber and Y. Weiss. Multibody factorization with uncertainity and missing data using the EM algorithm. *CVPR*, 1:707–714, 2004.

[9] J. Xiao and T. Kanade. Non-rigid shape and motion recovery: Degenerate deformations. *CVPR*, 1:668–675, 2004.

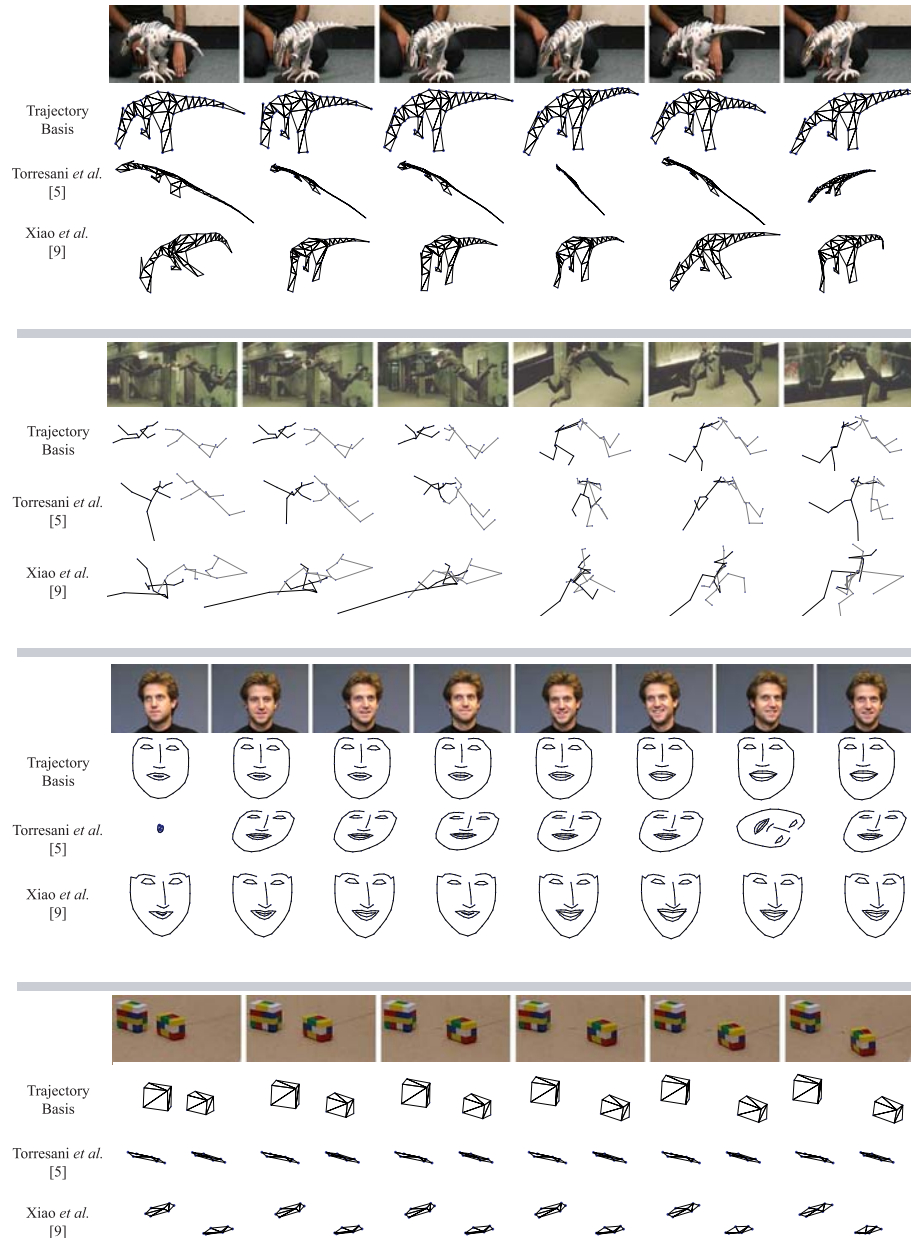

Figure 6: Results on Dinosaur, Matrix, PIE face, and Cubes sequences. $k$ was set to 12, 3, 2, and 2 respectively.

[10] M. Brand. Morphable 3D models from video. *CVPR*, 2:456, 2001.

[11] A. Del Bue, F.Smeraldi, and L. Agapito. Non-rigid structure from motion using ranklet-based tracking and non-linear optimization. *IVC*, pages 297–310, 2007.

[12] Amnon Shashua. Trilinear tensor: The fundamental construct of multiple-view geometry and its applications. *AFPAC*, 1997.

[13] Lihi Zelnik-Manor and Michal Irani. Temporal factorization vs. spatial factorization. *ECCV*, 2004.

[14] O. Arikan. Compression of motion capture databases. *ACM Trans. on Graphics*, 2006.

[15] A. Datta, Y. Sheikh, and T. Kanade. Linear motion estimation for systems of articulated planes. *CVPR*, 2008.
